# Gaussian Process Priors With Uncertain Inputs Application to Multiple-Step Ahead Time Series Forecasting

**Agathe Girard**
Department of Computing Science
University of Glasgow
Glasgow, G12 8QQ
*agathe@dcs.gla.ac.uk*

**Carl Edward Rasmussen**
Gatsby Unit
University College London
London, WC1N 3AR
*edward@gatsby.ucl.ac.uk*

**Joaquin Quiñonero Candela**
Informatics and Mathematical Modelling
Technical University of Denmark
Richard Petersens Plads, Building 321
DK-2800 Kongens, Lyngby, Denmark
*jqc@imm.dtu.dk*

**Roderick Murray-Smith**
Department of Computing Science
University of Glasgow, Glasgow, G12 8QQ
& Hamilton Institute
National University of Ireland, Maynooth
*rod@dcs.gla.ac.uk*

## Abstract

We consider the problem of multi-step ahead prediction in time series analysis using the non-parametric Gaussian process model. $k$-step ahead forecasting of a discrete-time non-linear dynamic system can be performed by doing repeated one-step ahead predictions. For a state-space model of the form $y_t = f(y_{t-1}, \ldots, y_{t-L})$, the prediction of $y$ at time $t + k$ is based on the point estimates of the previous outputs. In this paper, we show how, using an analytical Gaussian approximation, we can formally incorporate the uncertainty about intermediate regressor values, thus updating the uncertainty on the current prediction.

## 1 Introduction

One of the main objectives in time series analysis is forecasting and in many real life problems, one has to predict ahead in time, up to a certain time horizon (sometimes called *lead* time or prediction horizon). Furthermore, knowledge of the uncertainty of the prediction is important. Currently, the multiple-step ahead prediction task is achieved by either explic-

itly training a *direct* model to predict $k$ steps ahead, or by doing repeated one-step ahead predictions up to the desired horizon, which we call the *iterative method*.

There are a number of reasons why the iterative method might be preferred to the 'direct' one. Firstly, the direct method makes predictions for a fixed horizon only, making it computationally demanding if one is interested in different horizons. Furthermore, the larger $k$, the more training data we need in order to achieve a good predictive performance, because of the larger number of 'missing' data between $t$ and $t + k$. On the other hand, the iterated method provides any $k$-step ahead forecast, up to the desired horizon, as well as the joint probability distribution of the predicted points.

In the Gaussian process modelling approach, one computes predictive distributions whose means serve as output estimates. Gaussian processes (GPs) for regression have historically been first introduced by O'Hagan [1] but started being a popular non-parametric modelling approach after the publication of [7]. In [10], it is shown that GPs can achieve a predictive performance comparable to (if not better than) other modelling approaches like neural networks or local learning methods. We will show that for a $k$-step ahead prediction which ignores the accumulating prediction variance, the model is not conservative enough, with unrealistically small uncertainty attached to the forecast. An alternative solution is presented for iterative $k$-step ahead prediction, with propagation of the prediction uncertainty.

## 2 Gaussian Process modelling

We briefly recall some fundamentals of Gaussian processes. For a comprehensive introduction, please refer to [5], [11], or the more recent review [12].

### 2.1 The GP prior model

Formally, the random function, or stochastic process, $f(\mathbf{x})$ is a Gaussian process, with mean $m(\mathbf{x})$ and covariance function $C(\mathbf{x}^p, \mathbf{x}^q)$, if its values at a finite number of points, $f(\mathbf{x}^1), \ldots, f(\mathbf{x}^n)$, are seen as the components of a normally distributed random vector. If we further assume that the process is stationary: it has a constant mean and a covariance function only depending on the distance between the inputs $\mathbf{x}$. For any $n$, we have

$$f(\mathbf{x}^1), \ldots, f(\mathbf{x}^n) \sim \mathcal{N}(\mathbf{0}, \mathbf{\Sigma}) \,, \tag{1}$$

with $\Sigma_{pq} = \mathrm{Cov}(f(\mathbf{x}^p), f(\mathbf{x}^q)) = C(\mathbf{x}^p, \mathbf{x}^q)$ giving the covariance between the points $f(\mathbf{x}^p)$ and $f(\mathbf{x}^q)$, which is a function of the inputs corresponding to the same cases $p$ and $q$. A common choice of covariance function is the Gaussian kernel[1]

$$C(\mathbf{x}^p, \mathbf{x}^q) = \exp\left[ -\frac{1}{2} \sum_{d=1}^{D} \frac{(\mathbf{x}_d^p - \mathbf{x}_d^q)^2}{w_d^2} \right] \,, \tag{2}$$

where $D$ is the input dimension. The $w$ parameters (correlation length) allow a different distance measure for each input dimension $d$. For a given problem, these parameters will be adjusted to the data at hand and, for irrelevant inputs, the corresponding $w_d$ will tend to zero.

The role of the covariance function in the GP framework is similar to that of the kernels used in the Support Vector Machines community. This particular choice corresponds to a prior assumption that the underlying function $f$ is *smooth* and *continuous*. It accounts for a high correlation between the outputs of cases with nearby inputs.

## 2.2 Predicting with Gaussian Processes

Given this prior on the function $f$ and a set of data $\mathcal{D} = \{y^i, \mathbf{x}^i\}_{i=1}^N$, our aim, in this Bayesian setting, is to get the predictive distribution of the function value $f(\mathbf{x}^*)$ corresponding to a new (given) input $\mathbf{x}^*$.

If we assume an additive uncorrelated Gaussian white noise, with variance $v_0$, relates the targets (observations) to the function outputs, the distribution over the targets is Gaussian, with zero mean and covariance matrix such that $K_{pq} = \Sigma_{pq} + v_0 \delta_{pq}$. We then adjust the vector of hyperparameters $\Theta = [w_1 \dots w_D\ v_1\ v_0]^T$ so as to maximise the log-likelihood $\mathcal{L}(\Theta) = \log p(\mathbf{y}|\Theta)$, where $\mathbf{t}$ is the vector of observations.

In this framework, for a new $\mathbf{x}^*$, the predictive distribution is simply obtained by conditioning on the training data. The joint distribution of the variables being Gaussian, this conditional distribution, $p(f(\mathbf{x})^*|\mathbf{x}^*, \mathcal{D})$ is also Gaussian with mean and variance

$$\mu(\mathbf{x}^*) = \mathbf{k}(\mathbf{x}^*)^T \mathbf{K}^{-1} \mathbf{y} \tag{3}$$

$$\sigma^2(\mathbf{x}^*) = k(\mathbf{x}^*) - \mathbf{k}(\mathbf{x}^*)^T K^{-1} \mathbf{k}(\mathbf{x}^*), \tag{4}$$

where $\mathbf{k}(\mathbf{x}^*) = [C(\mathbf{x}^*, \mathbf{x}^1), \dots, C(\mathbf{x}^*, \mathbf{x}^N)]^T$ is the $N \times 1$ vector of covariances between the new point and the training targets and $k(\mathbf{x}^*) = C(\mathbf{x}^*, \mathbf{x}^*) = 1$, with $C(., .)$ as given by (2).

The predictive mean serves as a point estimate of the function output, $\hat{f}(\mathbf{x}^*)$ with uncertainty $\sigma(\mathbf{x}^*)$. And it is also a point estimate for the target, $\hat{y}^*$, with variance $\sigma^2(\mathbf{x}^*) + v_0$.

# 3 Prediction at a random input

If we now assume that the input distribution is Gaussian, $\mathbf{x}^* \sim \mathcal{N}(\mu_{x^*}, \Sigma_{x^*})$, the predictive distribution is now obtain by integrating over $\mathbf{x}^*$

$$p(f(\mathbf{x}^*)|\mu_{x^*}, \Sigma_{x^*}) = \int p(f(\mathbf{x}^*)|\mathbf{x}^*, \mathcal{D}) p(\mathbf{x}^*) d\mathbf{x}^*, \tag{5}$$

where $p(f(\mathbf{x}^*)|\mathbf{x}^*, \mathcal{D})$ is Normal, as specified by (3) and (4).

## 3.1 Gaussian approximation

Given that this integral is analytically intractable ($p(f(\mathbf{x}^*)|\mathbf{x}^*)$ is a complicated function of $\mathbf{x}^*$), we opt for an analytical Gaussian approximation and only compute the mean and variance of $p(f(\mathbf{x}^*)|\mu_{x^*}, \Sigma_{x^*})$. Using the law of iterated expectations and conditional variance, the 'new' mean and variance are given by

$$m(\mu_{x^*}, \Sigma_{x^*}) = E_{x^*}[E_{f(x^*)}[f(\mathbf{x}^*)|\mathbf{x}^*]] = E_{x^*}[\mu(\mathbf{x}^*)] \tag{6}$$

$$v(\mu_{x^*}, \Sigma_{x^*}) = E_{x^*}[\text{var}_{f(x^*)}(f(\mathbf{x}^*)|\mathbf{x}^*)] + \text{var}_{x^*}(E_{f(x^*)}[f(\mathbf{x}^*)|\mathbf{x}^*])$$

$$= E_{x^*}[\sigma^2(\mathbf{x}^*)] + \text{var}_{x^*}(\mu(\mathbf{x}^*)) \tag{7}$$

where $E_{x^*}$ indicates the expectation under $x^*$.

In our initial development, we made additional approximations ([2]). A first and second order Taylor expansions of $\mu(\mathbf{x}^*)$ and $\sigma^2(\mathbf{x}^*)$ respectively, around $\mu_{x^*}$, led to

$$m(\mu_{x^*}, \Sigma_{x^*}) = \mu(\mu_{x^*}) \tag{8}$$

$$v(\mu_{x^*}, \Sigma_{x^*}) = \sigma^2(\mu_{x^*}) + \frac{1}{2}\text{Tr}\left\{\left.\frac{\partial^2 \sigma^2(\mathbf{x}^*)}{\partial \mathbf{x}^* \partial \mathbf{x}^{*T}}\right|_{\mathbf{x}^*=\mu_{x^*}} \Sigma_{x^*}\right\} + \left.\frac{\partial \mu(\mathbf{x}^*)}{\partial \mathbf{x}^*}\right|^T_{\mathbf{x}^*=\mu_{x^*}} \Sigma_{x^*} \left.\frac{\partial \mu(\mathbf{x}^*)}{\partial \mathbf{x}^*}\right|_{\mathbf{x}^*=\mu_{x^*}} \tag{9}$$

The detailed calculations can be found in [2].

In [8], we derived the exact expressions of the first and second moments. Rewriting the predictive mean $\mu(\mathbf{x}^*)$ as a linear combination of the covariance between the new $\mathbf{x}^*$ and the training points (as suggested in [12]), with our choice of covariance function, the calculation of $m(\mathbf{x}^*)$ then involves the product of two Gaussian functions:

$$m(\mu_{x^*}, \Sigma_{x^*}) = \int \mu(\mathbf{x}^*) p(\mathbf{x}^*) d\mathbf{x}^* = \sum_j \beta_j \int C(\mathbf{x}^*, \mathbf{x}_j) p(\mathbf{x}^*) d\mathbf{x}^* \qquad (10)$$

with $\beta = \mathbf{K}^{-1} \mathbf{y}$. This leads to (refer to [9] for details)

$$m(\mu_{x^*}, \Sigma_{x^*}) = \mathbf{q}^{\mathbf{T}} \beta \qquad (11)$$

with $q_i = |\mathbf{W}^{-1}\Sigma_{x^*} + I|^{-1/2} \exp\left(-\frac{1}{2}(\mu_{x^*} - \mathbf{x_i})^T(\Sigma_{x^*} + \mathbf{W})^{-1}(\mu_{x^*} - \mathbf{x}_i)\right)$, where $\mathbf{W} = \mathrm{diag}[w_1^2, \ldots, w_D^2]$ and $I$ is the $D \times D$ identity matrix.

In the same manner, we obtain for the variance

$$v(\mu_{x^*}, \Sigma_{x^*}) = C(\mu_{x^*}, \mu_{x^*} + \mathrm{Tr}\left[(\beta\beta^{\mathbf{T}} - \mathbf{K}^{-1})Q\right] - \mathrm{Tr}(\mathbf{q}^{\mathbf{T}}\beta)^2 \qquad (12)$$

with

$$\begin{aligned}
Q_{ij} &= |2\mathbf{W}^{-1}\Sigma_{x^*} + I|^{-1/2} \exp\left(-\frac{1}{2}(\mathbf{xb} - \mu_{x^*})^T(\frac{1}{2}\mathbf{W} + \Sigma_{x^*})^{-1}(\mathbf{xb} - \mu_{x^*})\right) \\
&\quad \exp\left(-\frac{1}{2}(\mathbf{x_i} - \mathbf{x_j})^T(2\mathbf{W})^{-1}(\mathbf{x_i} - \mathbf{x_j})\right)
\end{aligned} \qquad (13)$$

where $\mathbf{xb} = (\mathbf{x}_i + \mathbf{x}_j)/2$.

### 3.2 Monte-Carlo alternative

Equation (5) can be solved by performing a numerical approximation of the integral, using a simple Monte-Carlo approach:

$$p(f(\mathbf{x}^*)|\mu_{x^*}, \Sigma_{x^*}) = \int p(f(\mathbf{x}^*)|\mathbf{x}^*)p(\mathbf{x}^*)d\mathbf{x}^* \simeq \frac{1}{T}\sum_{t=1}^{T} p(f(\mathbf{x}^*)|\mathbf{x}^{*t}), \qquad (14)$$

where $\mathbf{x}^{*t}$ are (independent) samples from $p(\mathbf{x}^*)$.

## 4 Iterative $k$-step ahead prediction of time series

For the multiple-step ahead prediction task of time series, the iterative method consists in making repeated one-step ahead predictions, up to the desired horizon. Consider the time series $y^{t_1}, \ldots, y^t$ and the state-space model $y^{t_i} = f(x^{t_i}) + \epsilon^{t_i}$ where $x^{t_i} = [y^{t_i-1}, \ldots, y^{t_i-L}]^T$ is the *state* at time $t_i$ (we assume that the lag $L$ is known) and the (white) noise has variance $v_0$.

Then, the"naive" iterative $k$-step ahead prediction method works as follows: it predicts only one time step ahead, using the estimate of the output of the current prediction, as well as previous outputs (up to the lag $L$), as the input to the prediction of the next time step, until the prediction $k$ steps ahead is made. That way, only the output estimates are used and the uncertainty induced by each successive prediction is not accounted for.

Using the results derived in the previous section, we suggest to formally incorporate the uncertainty information about the intermediate regressor. That is, as we predict ahead in time, we now view the lagged outputs as random variables. In this framework, the input

at time $t_k$ is a random vector with mean formed by the predicted means of the lagged outputs $y^{t+k-\tau}$, $\tau = 1, \ldots, L$, given by (11). The $L \times L$ input covariance matrix has the different predicted variances on its diagonal (with the estimated noise variance $v_0$ added to them), computed with (12), and the off-diagonal elements are given by, in the case of the exact solution, $\mathrm{cov}(y_{t_k}, x_{t_k}) = \sum_i \beta_i q_i (c_i - \mu_{x_{t_k}})$, where $q_i$ is as defined previously and $c_i = C(\mathbf{W}^{-1}\mathbf{x}_i + \mathbf{\Sigma}_{x^*}^{-1}\mu_{x^*})$ with $C = (\mathbf{W}^{-1} + \mathbf{\Sigma}_{x^*}^{-1})^{-1}$.

## 4.1 Illustrative examples

The first example is intended to provide a basis for comparing the approximate and exact solutions, within the Gaussian approximation of (5)), to the numerical solution (Monte-Carlo sampling from the true distribution), when the uncertainty is propagated as we predict ahead in time. We use the second example, inspired from real-life problems, to show that iteratively predicting ahead in time without taking account of the uncertainties induced by each succesive prediction leads to inaccurate results, with unrealistically small error bars.

We then assess the predictive performance of the different methods by computing the average absolute error ($L_1$), the average squared error ($L_2$) and average minus log predictive density[2] ($L_3$), which measures the density of the actual true test output under the Gaussian predictive distribution and use its negative log as a measure of loss.

### 4.1.1 Forecasting the Mackey-Glass time series

The Mackey-Glass chaotic time series constitutes a wellknown benchmark and a challenge for the multiple-step ahead prediction task, due to its strong non-linearity [4]: $\frac{dz(t)}{dt} = -bz(t) + a\frac{z(t-\tau)}{1+z(t-\tau)^{10}}$. We have $a = 0.2$, $b = 0.1$ and $\tau = 17$. The series is re-sampled with period 1 and normalized. We choose $L = 16$ for the number of lagged outputs in the state vector, $\mathbf{x_k} = [\mathbf{z_{k-1}}, \mathbf{z_{k-2}}, \ldots, \mathbf{z_{k-L}}]$ and the targets, $t_k = z_k$, are corrupted by a white noise with variance 0.001.

We train a GP model with a Gaussian kernel such as (2) on 100 points, taken at random from a series of 8000 points. Figure 1 shows the mean predictions with their uncertainties, given by the exact and approximate methods, and 50 samples from the Monte-Carlo numerical approximation, from $k = 1$ to $k = 100$ steps ahead, for different starting points. Figure 2 shows the plot of the 100-step ahead mean predictions (left) and their $2\sigma$ uncertainties (right), given by the exact and approximate methods, as well as the sample mean and sample variance obtained with the numerical solution (average over 50 points).

These figures show the better performance of the exact method on the approximate one. Also, they allow us to validate the Gaussian approximation, noticing that the error bars encompass the samples from the true distribution. Table 1 provides a quantitative confirmation.

Table 1: Average (over 500 test points) absolute error ($L_1$), squared error ($L_2$) and minus log predictive density ($L_3$) of the 100-step ahead predictions obtained using the exact method ($M_1$), the approximate one ($M_2$) and the sampling from the true distribution ($M_3$).

|       | $L_1$  | $L_2$  | $L_3$  |
|-------|--------|--------|--------|
| $M_1$ | 0.4574 | 0.3397 | 0.8473 |
| $M_2$ | 0.5409 | 0.4609 | 1.1300 |
| $M_3$ | 0.4886 | .3765  | 0.8979 |

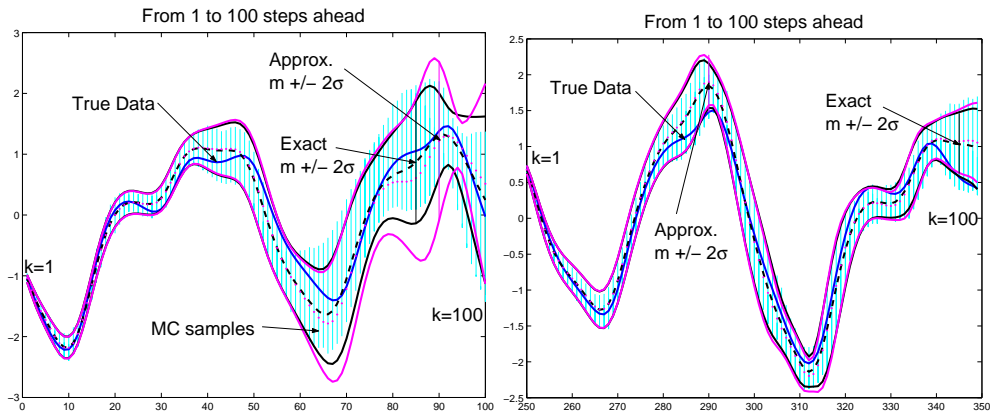

Figure 1: Iterative method in action: simulation from 1 to 100 steps ahead for different starting points in the test series. Mean predictions with $2\sigma$ error bars given by the exact (dash) and approximate (dot) methods. Also plotted, $50$ samples obtained using the numerical approximation.

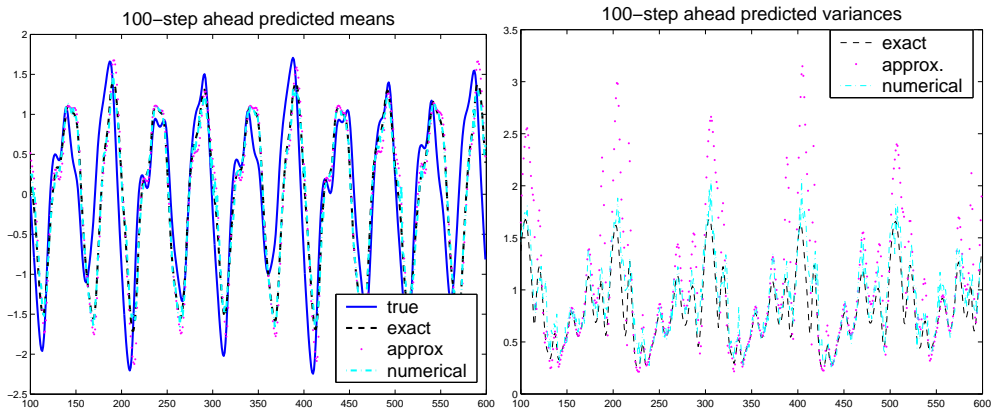

Figure 2: 100-step ahead mean predictions (left) and uncertainties (right.) obtained using the exact method (dash), the approximate (dot) and the sample mean and variance of the numerical solution (dash-dot).

#### 4.1.2 Prediction of a pH process simulation

We now compare the iterative $k$-step ahead prediction results obtained when propagating the uncertainty (using the approximate method) and when using the output estimates only (the naive approach). For doing so, we use the pH neutralisation process benchmark presented in [3]. The training and test data consist of pH values (outputs $y$ of the process) and a control input signal ($u$).

With a model of the form $y_t = f(y_{t-8}, \ldots, y_{t-1}, u_{t-8}, \ldots, u_{t-1})$, we train our GP on 1228 examples and consider a test set of ?? points (all data have been normalized).

Figure 3 shows the 10-step ahead predicted means and variances obtained when propagating the uncertainty and when using information on the past predicted means only. The losses calculated are the following: $L_1 = 0.1716$, $L_2 = 0.0574$ and $L_3 = 0.6208$ for the approximate method and $L_3 = 1980.2$ for the naive one!

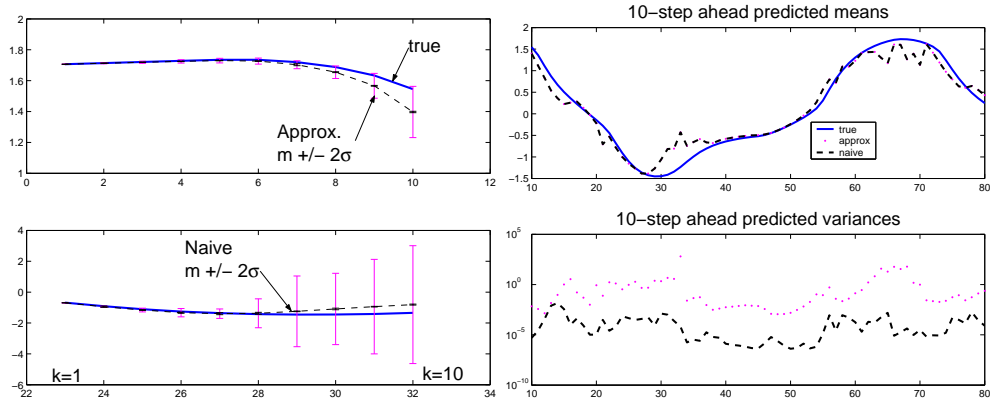

Figure 3: Predictions from 1 to 10 steps ahead (left). 10-step ahead mean predictions with the corresponding variances, when propagating the uncertainty (dot) and when using the previous point estimates only (dash).

## 5 Conclusions

We have presented a novel approach which allows us to use knowledge of the variance on inputs to Gaussian process models to achieve more realistic prediction variance in the case of noisy inputs.

Iterating this approach allows us to use it as a method for efficient propagation of uncertainty in the multi-step ahead prediction task of non-linear time-series. In experiments on simulated dynamic systems, comparing our Gaussian approximation to Monte Carlo simulations, we found that the propagation method is comparable to Monte Carlo simulations, and that both approaches achieved more realistic error bars than a naive approach which ignores the uncertainty on current state.

This method can help understanding the underlying dynamics of a system, as well as being useful, for instance, in a model predictive control framework where knowledge of the accuracy of the model predictions over the whole prediction horizon is required (see [6] for a model predictive control law based on Gaussian processes taking account of the prediction uncertainty). Note that this method is also useful in its own right in the case of noisy model inputs, assuming they have a Gaussian distribution.

## Acknowledgements

Many thanks to Mike Titterington for his useful comments. The authors gratefully acknowledge the support of the *Multi-Agent Control* Research Training Network - EC TMR grant HPRN-CT-1999-00107 and RM-S is grateful for EPSRC grant *Modern statistical approaches to off-equilibrium modelling for nonlinear system control* GR/M76379/01.

## Footnotes

[1]This choice was motivated by the fact that, in [8], we were aiming at unified expressions for the GPs and the Relevance Vector Machines models which employ such a kernel. More discussion about possible covariance functions can be found in [5].

[2] To evaluate these losses in the case of Monte-Carlo sampling, we use the sample mean and sample variance.

## References

[1] O'Hagan, A. (1978) Curve fitting and optimal design for prediction. *Journal of the Royal Statistical Society B* **40**:1-42.

[2] Girard, A. & Rasmussen, C. E. & Murray-Smith, R. (2002) Gaussian Process Priors With Uncertain Inputs: Multiple-Step Ahead Prediction. Technical Report, TR-2002-119, Dept. of Computing Science, University of Glasgow.

[3] Henson, M. A. & Seborg, D. E. (1994) Adaptive nonlinear control of a pH neutralisation process. *IEEE Trans Control System Technology* **2**:169-183.

[4] Mackey, M. C. & Glass, L. (1977) Oscillation and Chaos in Physiological Control Systems. *Science* **197**:287-289.

[5] MacKay, D. J. C. (1997) Gaussian Processes - A Replacement for Supervised Neural Networks?. Lecture notes for a tutorial at NIPS 1997.

[6] Murray-Smith, R. & Sbarbaro-Hofer, D. (2002) Nonlinear adaptive control using non-parametric Gaussian process prior models. *15th IFAC World Congress on Automatic Control, Barcelona*

[7] Neal, R. M. (1995) *Bayesian Learning for Neural Networks* PhD thesis, Dept. of Computer Science, University of Toronto.

[8] Quiñonero Candela, J & Girard, A. & Larsen, J. (2002) Propagation of Uncertainty in Bayesian Kernels Models – Application to Multiple-Step Ahead Forecasting *Submitted to ICASSP 2003*.

[9] Quiñonero Candela, J. & Girard, A. (2002) Prediction at an Uncertain Input for Gaussian Processes and Relevance Vector Machines - Application to Multiple-Step Ahead Time-Series Forecasting. Technical Report, IMM, Danish Technical University.

[10] Rasmussen, C. E. (1996) *Evaluation of Gaussian Processes and other Methods for Non-Linear Regression* PhD thesis, Dept. of Computer Science, University of Toronto.

[11] Williams, C. K. I. & Rasmussen, C. E. (1996) Gaussian Processes for Regression *Advances in Neural Information Processing Systems 8* MIT Press.

[12] Williams, C. K. I. (2002) Gaussian Processes *To appear in The handbook of Brain Theory and Neural Networks, Second edition* MIT Press.
